# Metric Learning by Collapsing Classes

**Amir Globerson**
School of Computer Science and Engineering,
Interdisciplinary Center for Neural Computation
The Hebrew University Jerusalem, 91904, Israel
gamir@cs.huji.ac.il

**Sam Roweis**
Machine Learning Group
Department of Computer Science
University of Toronto, Canada
roweis@cs.toronto.edu

## Abstract

We present an algorithm for learning a quadratic Gaussian metric (Mahalanobis distance) for use in classification tasks. Our method relies on the simple geometric intuition that a good metric is one under which points in the same class are simultaneously near each other and far from points in the other classes. We construct a convex optimization problem whose solution generates such a metric by trying to collapse all examples in the same class to a single point and push examples in other classes infinitely far away. We show that when the metric we learn is used in simple classifiers, it yields substantial improvements over standard alternatives on a variety of problems. We also discuss how the learned metric may be used to obtain a compact low dimensional feature representation of the original input space, allowing more efficient classification with very little reduction in performance.

## 1 Supervised Learning of Metrics

The problem of learning a distance measure (metric) over an input space is of fundamental importance in machine learning [10, 9], both supervised and unsupervised. When such measures are learned directly from the available data, they can be used to improve learning algorithms which rely on distance computations such as nearest neighbour classification [5], supervised kernel machines (such as GPs or SVMs) and even unsupervised clustering algorithms [10]. Good similarity measures may also provide insight into the underlying structure of data (e.g. inter-protein distances), and may aid in building better data visualizations via embedding. In fact, there is a close link between distance learning and feature extraction since whenever we construct a feature $f(x)$ for an input space $X$, we can measure distances between $x_1, x_2 \in X$ using a simple distance function (e.g. Euclidean) $d[f(x_1), f(x_2)]$ in feature space. Thus by fixing $d$, any feature extraction algorithm may be considered a metric learning method. Perhaps the simplest illustration of this approach is when the $f(x)$ is a linear projection of $\mathbf{x} \in \Re^r$ so that $f(\mathbf{x}) = W\mathbf{x}$. The Euclidean distance between $f(\mathbf{x}_1)$ and $f(\mathbf{x}_2)$ is then the Mahalanobis distance $\|f(\mathbf{x}_1) - f(\mathbf{x}_2)\|^2 = (\mathbf{x}_1 - \mathbf{x}_2)^T A(\mathbf{x}_1 - \mathbf{x}_2)$, where $A = W^T W$ is a positive semidefinite matrix. Much of the recent work on metric learning has indeed focused on learning Mahalanobis distances, i.e. learning the matrix $A$. This is also the goal of the current work.

A common approach to learning metrics is to assume some knowledge in the form of equiv-

alence relations, i.e. which points should be close and which should be far (without specifying their exact distances). In the classification setting there is a natural equivalence relation, namely whether two points are in the same class or not. One of the classical statistical methods which uses this idea for the Mahalanobis distance is Fisher's Linear Discriminant Analysis (see e.g. [6]). Other more recent methods are [10, 9, 5] which seek to minimize various separation criteria between the classes under the new metric.

In this work, we present a novel approach to learning such a metric. Our approach, the Maximally Collapsing Metric Learning algorithm (MCML), relies on the simple geometric intuition that if all points in the same class could be mapped into a single location in feature space and all points in other classes mapped to other locations, this would result in an ideal approximation of our equivalence relation. Our algorithm approximates this scenario via a stochastic selection rule, as in Neighborhood Component Analysis (NCA) [5]. However, unlike NCA, the optimization problem is convex and thus our method is completely specified by our objective function. Different initialization and optimization techniques may affect the speed of obtaining the solution but the final solution itself is unique. We also show that our method approximates the local covariance structure of the data, as opposed to Linear Discriminant Analysis methods which use only global covariance structure.

## 2 The Approach of Collapsing Classes

Given a set of $n$ labeled examples $(\mathbf{x}_i, y_i)$, where $\mathbf{x}_i \in \Re^r$ and $y_i \in \{1 \ldots k\}$, we seek a similarity measure between two points in $X$ space. We focus on Mahalanobis form metrics

$$d(\mathbf{x}_i, \mathbf{x}_j | A) = d_{ij}^A = (\mathbf{x}_i - \mathbf{x}_j)^T A (\mathbf{x}_i - \mathbf{x}_j) , \tag{1}$$

where $A$ is a positive semidefinite (PSD) matrix.

Intuitively, what we want from a good metric is that it makes elements of $X$ in the same class look *close* whereas those in different classes appear *far*. Our approach starts with the ideal case when this is true in the most optimistic sense: same class points are at zero distance, and different class points are infinitely far. Alternatively this can be viewed as mapping $\mathbf{x}$ via a linear projection $W\mathbf{x}$ ($A = W^T W$), such that all points in the same class are mapped into the same point. This intuition is related to the analysis of spectral clustering [8], where the ideal case analysis of the algorithm results in all same cluster points being mapped to a single point.

To learn a metric which approximates the ideal geometric setup described above, we introduce, for each training point, a conditional distribution over other points (as in [5]). Specifically, for each $\mathbf{x}_i$ we define a conditional distribution over points $i \neq j$ such that

$$p^A(j|i) = \frac{1}{Z_i} e^{-d_{ij}^A} = \frac{e^{-d_{ij}^A}}{\sum_{k \neq i} e^{-d_{ik}^A}} \qquad i \neq j . \tag{2}$$

If all points in the same class were mapped to a single point and infinitely far from points in different classes, we would have the ideal "bi-level" distribution:

$$p_0(j|i) \propto \left\{ \begin{array}{ll} 1 & y_i = y_j \\ 0 & y_i \neq y_j . \end{array} \right. \tag{3}$$

Furthermore, under very mild conditions, any set of points which achieves the above distribution must have the desired geometry. In particular, assume there are at least $\hat{r} + 2$ points in each class, where $\hat{r} = \text{rank}[A]$ (note that $\hat{r} \leq r$). Then $p^A(j|i) = p_0(j|i)$ ($\forall i, j$) implies that under $A$ all points in the same class will be mapped to a single point, infinitely far from other class points [1].

Thus it is natural to seek a matrix $A$ such that $p^A(j|i)$ is as close as possible to $p_0(j|i)$. Since we are trying to match distributions, we minimize the KL divergence $KL[p_0|p]$:

$$\min_A \sum_i KL[p_0(j|i)|p^A(j|i)] \qquad s.t.\ A \in PSD \qquad (4)$$

The crucial property of this optimization problem is that it is *convex* in the matrix $A$. To see this, first note that any convex linear combination of feasible solutions $A = \alpha A_0 + (1 - \alpha)A_1$ s.t. $0 \leq \alpha \leq 1$ is still a feasible solution, since the set of PSD matrices is convex. Next, we can show that $f(A)$ alway has a greater cost than either of the endpoints. To do this, we rewrite the objective function $f(A) = \sum_i KL[p_0(j|i)|p(j|i)]$ in the form [2]:

$$f(A) = -\sum_{i,j:y_j=y_i} \log p(j|i) = \sum_{i,j:y_j=y_i} d_{ij}^A + \sum_i \log Z_i$$

where we assumed for simplicity that classes are equi-probable, yielding a multiplicative constant. To see why $f(A)$ is convex, first note that $d_{ij}^A = (\mathbf{x}_i - \mathbf{x}_j)^T A(\mathbf{x}_i - \mathbf{x}_j)$ is linear in $A$, and thus convex. The function $\log Z_i$ is a $\log \sum \exp$ function of affine functions of $A$ and is therefore also convex (see [4], page 74).

## 2.1  Convex Duality

Since our optimization problem is convex, it has an equivalent convex dual. Specifically, the convex dual of Eq. (4) is the following entropy maximization problem:

$$\max_{p(j|i)} \quad \sum_i H[p(j|i)] \qquad s.t. \quad \sum_i E_{p_0(j|i)}[\mathbf{v}_{ji}\mathbf{v}_{ji}^T] - \sum_i E_{p(j|i)}[\mathbf{v}_{ji}\mathbf{v}_{ji}^T] \succeq 0 \qquad (5)$$

where $\mathbf{v}_{ji} = \mathbf{x}_j - \mathbf{x}_i$, $H[\cdot]$ is the entropy function and we require $\sum_j p(j|i) = 1\ \forall i$.

To prove this duality we start with the proposed dual and obtain the original problem in Equation 4 as its dual. Write the Lagrangian for the above problem (where $\lambda$ is PSD) [3]

$$L(p, \lambda, \beta) = -\sum_i H(p(j|i)) - Tr(\lambda(\sum_i (E_{p_0}[\mathbf{v}_{ji}\mathbf{v}_{ji}^T] - E_p[\mathbf{v}_{ji}\mathbf{v}_{ji}^T]))) - \sum_i \beta_i(\sum_j p(j|i) - 1)$$

The dual function is defined as $g(\lambda, \beta) = \min_p L(p, \lambda, \beta)$. To derive it, we first solve for the minimizing $p$ by setting the derivative of $L(p, \lambda, \beta)$ w.r.t. $p(j|i)$ equal to zero.

$$0 = 1 + \log p(j|i) + Tr(\lambda \mathbf{v}_{ji}\mathbf{v}_{ji}^T) - \beta_i \quad \Rightarrow \quad p(j|i) = e^{\beta_i - 1 - Tr(\lambda \mathbf{v}_{ji}\mathbf{v}_{ji}^T)}$$

Plugging this solution to $L(p, \lambda, \beta)$ we get $g(\lambda, \beta) = -Tr(\lambda \sum_i E_{p_0}[\mathbf{v}_{ji}\mathbf{v}_{ji}^T]) + \sum_i \beta_i - \sum_{i,j} p(j|i)$. The dual problem is to maximize $g(\lambda, \beta)$. We can do this analytically w.r.t. $\beta_i$, yielding $1 - \beta_i = \log \sum_j e^{-Tr(\lambda \mathbf{v}_{ji}\mathbf{v}_{ji}^T)}$.

Now note that $Tr(\lambda \mathbf{v}_{ji}\mathbf{v}_{ji}^T) = \mathbf{v}_{ji}^T \lambda \mathbf{v}_{ji} = d_{ji}^\lambda$, so we can write

$$g(\lambda) = -\sum_{i,j:y_i=y_j} d_{ji}^\lambda - \sum_i \log \sum_j e^{-d_{ji}^\lambda}$$

which is minus our original target function. Since $g(\lambda)$ should be maximized, and $\lambda \succeq 0$ we have the desired duality result (identifying $\lambda$ with $A$).

---

$p_0(j|i) = 0$ when $y_j \neq y_i$. For a given point $\mathbf{x}_i$, all the points $j$ in its class satisfy $p(j|i) \propto 1$. Due to the structure of $p(j|i)$ in Equation 2, and because it is obeyed for all points in $\mathbf{x}_i's$ class, this implies that all the points in that class are equidistant from each other. However, it is easy to show that the maximum number of *different* equidistant points (also known as the equilateral dimension [1]) in $\hat{r}$ dimensions is $\hat{r} + 1$. Since by assumption we have at least $\hat{r} + 2$ points in the class of $\mathbf{x}_i$, and $A$ maps points into $\Re^{\hat{r}}$, it follows that all points are identical.

[2]Up to an additive constant $-\sum_i H[p_0(j|i)]$.

[3]We consider the equivalent problem of minimizing minus entropy.

### 2.1.1 Relation to covariance based and embedding methods

The convex dual derived above reveals an interesting relation to covariance based learning methods. The sufficient statistics used by the algorithm are a set of $n$ "spread" matrices. Each matrix is of the form $E_{p_0(j|i)}[\mathbf{v}_{ji}\mathbf{v}_{ji}^T]$. The algorithm tries to find a maximum entropy distribution which matches these matrices when averaged over the sample.

This should be contrasted with the covariance matrices used in metric learning such as Fisher's Discriminant Analysis. The latter uses the within and between class covariance matrices. The within covariance matrix is similar to the covariance matrix used here, but is calculated with respect to the class means, whereas here it is calculated separately for every point, and is centered on this point. This highlights the fact that MCML is not based on Gaussian assumptions where it is indeed sufficient to calculate a single class covariance.

Our method can also be thought of as a supervised version of the Stochastic Neighbour Embedding algorithm [7] in which the "target" distribution is $p_0$ (determined by the class labels) and the embedding points are not completely free but are instead constrained to be of the form $W\mathbf{x}_i$.

## 2.2 Optimizing the Convex Objective

Since the optimization problem in Equation 4 is convex, it is guaranteed to have only a single minimum which is the globally optimal solution[4]. It can be optimized using any appropriate numerical convex optimization machinery; all methods will yield the same solution although some may be faster than others. One standard approach is to use interior point Newton methods. However, these algorithms require the Hessian to be calculated, which would require $O(d^4)$ resources, and could be prohibitive in our case. Instead, we have experimented with using a first order gradient method, specifically the projected gradient approach as in [10]. At each iteration we take a small step in the direction of the negative gradient of the objective function[5], followed by a projection back onto the PSD cone. This projection is performed simply by taking the eigen-decomposition of $A$ and removing the components with negative eigenvalues. The algorithm is summarized below:

---

**Input:** Set of labeled data points $(\mathbf{x}_i, y_i)$, $i = 1 \ldots n$

**Output:** PSD metric which optimally *collapses* classes.

**Initialization:** Initialize $A_0$ to some PSD matrix
(randomly or using some initialization heuristic).

**Iterate:**

- Set $A_{t+1} = A_t - \epsilon \bigtriangledown f(A_t)$ where
$\bigtriangledown f(A) = \sum_{ij}(p_0(j|i) - p(j|i))(\mathbf{x}_j - \mathbf{x}_i)(\mathbf{x}_j - \mathbf{x}_i)^T$
- Calculate the eigen-decomposition of $A_{t+1}$
$A_{t+1} = \sum_k \lambda_k \mathbf{u}_k \mathbf{u}_k^T$, then set $A_{t+1} = \sum_k \max(\lambda_k, 0)\mathbf{u}_k \mathbf{u}_k^T$

---

Of course in principle it is possible to optimize over the dual instead of the primal but in our case, if the training data consists of $n$ points in $r$-dimensional space then the primal has only $O(r^2/2)$ variables while the dual has $O(n^2)$ so it will almost always be more efficient to operate on the primal $A$ directly. One exception to this case may be the kernel version (Section 4) where the primal is also of size $O(n^2)$.

# 3 Low Dimensional Projections for Feature Extraction

The Mahalanobis distance under a metric $A$ can be interpreted as a linear projection of the original inputs by the square root of $A$, followed by Euclidean distance in the projected space. Matrices $A$ which have less than full rank correspond to Mahalanobis distances based on low dimensional projections. Such metrics and the induced distances can be advantageous for several reasons [5]. First, low dimensional projections can substantially reduce the storage and computational requirements of a supervised method since only the projections of the training points must be stored and the manipulations at test time all occur in the lower dimensional feature space. Second, low dimensional projections re-represent the inputs, allowing for a supervised embedding or visualization of the original data.

If we consider matrices $A$ with rank at most $q$, we can always represent them in the form $A = W^T W$ for some projection matrix $W$ of size $q \times r$. This corresponds to projecting the original data into a $q$-dimensional space specified by the rows of $W$. However, rank constraints on a matrix are not convex [4], and hence the rank constrained problem is not convex and is likely to have local minima which make the optimization difficult and ill-defined since it becomes sensitive to initial conditions and choice of optimization method.

Luckily, there is an alternative approach to obtaining low dimensional projections, which *does* specify a unique solution by sequentially solving two globally tractable problems. This is the approach we follow here. First we solve for a (potentially) full rank metric $A$ using the convex program outlined above, and then obtain a low rank projection from it via spectral decomposition. This is done by diagonalizing $A$ into the form $A = \sum_{i=1}^{r} \lambda_i \mathbf{v}_i \mathbf{v}_i^T$ where $\lambda_1 \geq \lambda_2 \ldots \lambda_r$ are eigenvalues of $A$ and $\mathbf{v}_i$ are the corresponding eigenvectors. To obtain a low rank projection we constrain the sum above to include only the $q$ terms corresponding to the $q$ largest eigenvalues: $A_q = \sum_{i=1}^{q} \lambda_i \mathbf{v}_i \mathbf{v}_i^T$. The resulting projection is uniquely defined (up to an irrelevant unitary transformation) as $W = \mathrm{diag}(\sqrt{\lambda_1}, \ldots \sqrt{\lambda_q})[\mathbf{v}_1^T; \ldots; \mathbf{v}_q^T]$.

In general, the projection returned by this approach is not guaranteed to be the same as the projection corresponding to minimizing our objective function subject to a rank constraint on $A$ unless the optimal metric $A$ is of rank less than or equal to $q$. However, as we show in the experimental results, it is often the case that for practical problems the optimal $A$ has an eigen-spectrum which is rapidly decaying, so that many of its eigenvalues are indeed very small, suggesting the low rank solution will be close to optimal.

# 4 Learning Metrics with Kernels

It is interesting to consider the case where $\mathbf{x}_i$ are mapped into a high dimensional feature space $\phi(\mathbf{x}_i)$ and a Mahalanobis distance is sought in this space. We focus on the case where dot products in the feature space may be expressed via a kernel function, such that $\phi(\mathbf{x}_i) \cdot \phi(\mathbf{x}_j) = k(\mathbf{x}_i, \mathbf{x}_j)$ for some kernel $k$. We now show how our method can be changed to accommodate this setting, so that optimization depends only on dot products.
Consider the regularized target function:

$$f_{Reg}(A) = \sum_i \mathrm{KL}[p_0(j|i)|p(j|i)] + \lambda Tr(A) , \tag{6}$$

where the regularizing factor is equivalent to the Frobenius norm of the projection matrix $W$ since $Tr(A) = \|W\|^2$. Deriving w.r.t. $W$ we obtain $W = UX$, where $U$ is some matrix which specifies $W$ as a linear combination of sample points, and the $i^{th}$ row of the matrix $X$ is $\mathbf{x}_i$. Thus $A$ is given by $A = X^T U^T U X$. Defining the PSD matrix $\hat{A} = U^T U$, we can recast our optimization as looking for a PSD matrix $\hat{A}$, where the Mahalanobis distance is $(\mathbf{x}_i - \mathbf{x}_j)^T X^T \hat{A} X (\mathbf{x}_i - \mathbf{x}_j) = (\mathbf{k}_i - \mathbf{k}_j)^T \hat{A} (\mathbf{k}_i - \mathbf{k}_j)$, where we define $\mathbf{k}_i = X \mathbf{x}_i$.

This is exactly our original distance, with $\mathbf{x}_i$ replaced by $\mathbf{k}_i$, which depends only on dot products in $X$ space. The regularization term also depends solely on the dot products since $Tr(A) = Tr(X^T \hat{A} X) = Tr(X X^T \hat{A}) = Tr(K \hat{A})$, where $K$ is the kernel matrix given by $K = X X^T$. Note that the trace is a linear function of $\hat{A}$, keeping the problem convex. Thus, as long as dot products can be represented via kernels, the optimization can be carried out without explicitly using the high dimensional space.

To obtain a low dimensional solution, we follow the approach in Section 3: obtain a decomposition $A = V^T D V$ [6], and take the projection matrix to be the first $q$ rows of $D^{0.5} V$. As a first step, we calculate a matrix $B$ such that $\hat{A} = B^T B$, and thus $A = X^T B^T B X$. Since $A$ is a correlation matrix for the rows of $B X$ it can be shown (as in Kernel PCA) that its (left) eigenvectors are linear combinations of the rows of $B X$. Denoting by $V = \boldsymbol{\alpha} B X$ the eigenvector matrix, we obtain, after some algebra, that $\boldsymbol{\alpha} B K B^T = D \boldsymbol{\alpha}$. We conclude that $\boldsymbol{\alpha}$ is an eigenvector of the matrix $B K B^T$. Denote by $\hat{\boldsymbol{\alpha}}$ the matrix whose rows are orthonormal eigenvectors of $B K B^T$. Then $V$ can be shown to be orthonormal if we set $V = D^{-0.5} \hat{\boldsymbol{\alpha}} B X$. The final projection will then be $D^{0.5} V \mathbf{x}_i = \hat{\boldsymbol{\alpha}} B \mathbf{k}_i$. Low dimensional projections will be obtained by keeping only the first $q$ components of this projection.

## 5 Experimental Results

We compared our method to several metric learning algorithms on a supervised classification task. Training data was first used to learn a metric over the input space. Then this metric was used in a 1-nearest-neighbor algorithm to classify a test set. The datasets we investigated were taken from the UCI repository and have been used previously in evaluating supervised methods for metric learning [10, 5]. To these we added the USPS handwritten digits (downsampled to 8x8 pixels) and the YALE faces [2] (downsampled to 31x22).

The algorithms used in the comparative evaluation were

- Fisher's Linear Discriminant Analysis (LDA), which projects on the eigenvectors of $S_W^{-1} S_B$ where $S_W, S_B$ are the within and between class covariance matrices.
- The method of Xing et al [10] which minimizes the mean *within class* distance, while keeping the mean *between class* distance larger than one.
- Principal Component Analysis (PCA). There are several possibilities for scaling the PCA projections. We tested several, and report results of the empirically superior one (PCAW), which scales the projection components so that the covariance matrix after projection is the identity. PCAW often performs poorly on high dimensions, but globally outperforms all other variants.

We also evaluated the kernel version of MCML with an RBF kernel (denoted by KMCML)[7]. Since all methods allow projections to lower dimensions we compared performance for different projection dimensions [8].

The out-of sample performance results (based on 40 random splits of the data taking 70% for training and 30% for testing[9]) are shown in Figure 1. It can be seen that when used in a simple nearest-neighbour classifier, the metric learned by MCML almost always performs as well as, or significantly better than those learned by all other methods, across most dimensions. Furthermore, the kernel version of MCML outperforms the linear one on most datasets.

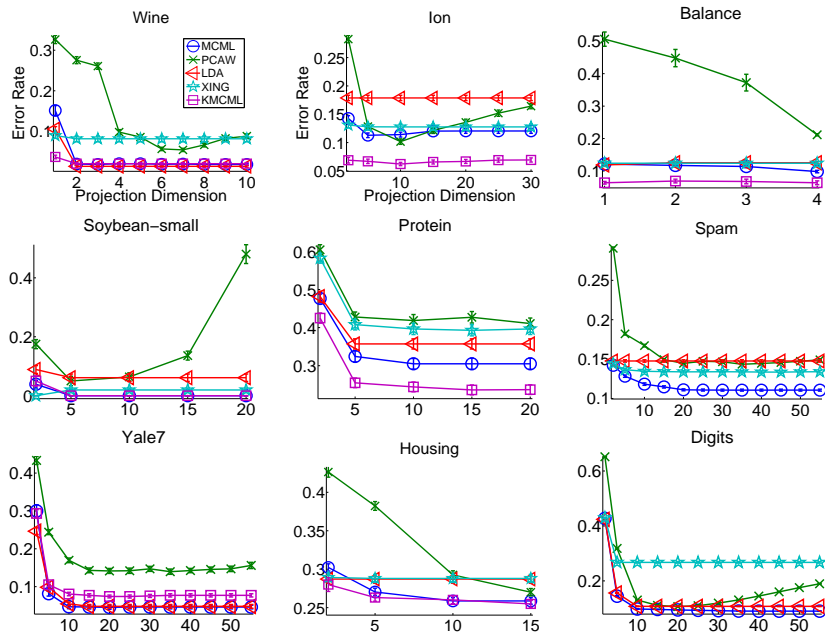

Figure 1: Classification error rate on several UCI datasets, USPS digits and YALE faces, for different projection dimensions. Algorithms are our Maximally Collapsing Metric Learning (MCML), Xing et.al.[10], PCA with whitening transformation (PCAW) and Fisher's Discriminant Analysis (LDA). Standard errors of the means shown on curves. No results given for XING on YALE and KMCML on Digits and Spam due to the data size.

## 5.1  Comparison to non convex procedures

The methods in the previous comparison are all well defined, in the sense that they are not susceptible to local minima in the optimization. They also have the added advantage of obtaining projections to all dimensions using one optimization run. Below, we also compare the MCML results to the results of two non-convex procedures. The first is the Non Convex variant of MCML (NMCML): The objective function of MCML can be optimized w.r.t the projection matrix $W$, where $A = W^T W$. Although this is no longer a convex problem, it is not constrained and is thus easier to optimize. The second non convex method is Neighbourhood Components Analysis (NCA) [5], which attempts to directly minimize the error incurred by a nearest neighbor classifier.

For both methods we optimized the matrix $W$ by restarting the optimization separately for each size of $W$. Minimization was performed using a conjugate gradient algorithm, initialized by LDA or randomly. Figure 2 shows results on a subset of the UCI datasets. It can be seen that the performance of NMCML is similar to that of MCML, although it is less stable, possibly due to local minima, and both methods usually outperform NCA. The inset in each figure shows the spectrum of the MCML matrix $A$, revealing that it often drops quickly after a few dimensions. This illustrates the effectiveness of our two stage optimization procedure, and suggests its low dimensional solutions are close to optimal.

## 6  Discussion and Extensions

We have presented an algorithm for learning maximally collapsing metrics (MCML), based on the intuition of collapsing classes into single points. MCML assumes that each class

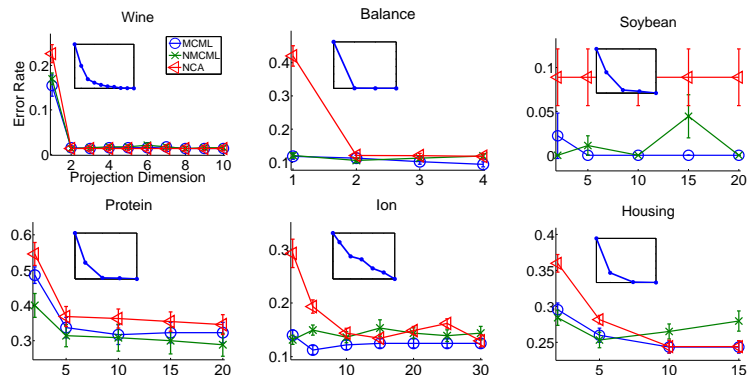

Figure 2: Classification error for non convex procedures, and the MCML method. Eigen-spectra for the MCML solution are shown in the inset.

may be collapsed to a single point, at least approximately, and thus is only suitable for uni-modal class distributions (or for simply connected sets if kernelization is used). However, if points belonging to a single class appear in several disconnected clusters in input (or feature) space, it is unlikely that MCML could collapse the class into a single point. It is possible that using a mixture of distributions, an EM-like algorithm can be constructed to accommodate this scenario.

The method can also be used to learn low dimensional projections of the input space. We showed that it performs well, even across a range of projection dimensions, and consistently outperforms existing methods. Finally, we have shown how the method can be extended to projections in high dimensional feature spaces using the kernel trick. The resulting nonlinear method was shown to improve classification results over the linear version.

## Footnotes

[1]Proof sketch: The infinite separation between points of different classes follows simply from

[4]When the data can be exactly collapsed into single class points, there will be multiple solutions at infinity. However, this is very unlikely to happen in real data.

[5]In the experiments, we used an Armijo like step size rule, as described in [3].

[6] Where $V$ is orthonormal, and the eigenvalues in $D$ are sorted in decreasing order.

[7] The regularization parameter $\lambda$ and the width of the RBF kernel were chosen using 5 fold cross-validation. KMCML was only evaluated for datasets with less than 1000 training points.

[8] To obtain low dimensional mappings we used the approach outlined in Section 3.

[9] Except for the larger datasets where 1000 random samples were used for training.

## References

[1] N. Alon and P. Pudlak. Equilateral sets in $l_p^n$. *Geom. Funct. Anal.*, 13(3), 2003.

[2] P. N. Belhumeur, J. Hespanha, and D. J. Kriegman. Eigenfaces vs. Fisherfaces: Recognition using class specific linear projection. In *ECCV (1)*, 1996.

[3] D.P. Bertsekas. On the Goldstein-Levitin-Polyak gradient projection method. *IEEE Transaction on Automatic Control*, 21(2):174–184, 1976.

[4] S. Boyd and L. Vandenberghe. *Convex Optimization*. Cambridge Univ. Press, 2004.

[5] J. Goldberger, S. Roweis, G. Hinton, and R. Salakhutdinov. Neighbourhood components analysis. In *Advances in Neural Information Processing Systems (NIPS)*, 2004.

[6] T. Hastie, R. Tibshirani, and J.H. Friedman. *The elements of statistical learning: data mining, inference, and prediction*. New York: Springer-Verlag, 2001.

[7] G. Hinton and S. Roweis. Stochastic neighbor embedding. In *Advances in Neural Information Processing Systems (NIPS)*, 2002.

[8] A. Ng, M. Jordan, and Y. Weiss. On spectral clustering: Analysis and an algorithm. In *Advances in Neural Information Processing Systems (NIPS)*, 2001.

[9] N. Shental, T. Hertz, D. Weinshall, and M. Pavel. Adjustment learning and relevant component analysis. In *Proc. of ECCV*, 2002.

[10] E. Xing, A. Ng, M. Jordan, and S. Russell. Distance metric learning, with application to clustering with side-information. In *Advances in Neural Information Processing Systems (NIPS)*, 2004.
